# Inference and communication in the game of Password

**Yang Xu*** and **Charles Kemp**[†]
Machine Learning Department*
School of Computer Science*
Department of Psychology[†]
Carnegie Mellon University
{yxl@cs.cmu.edu, ckemp@cmu.edu}

## Abstract

Communication between a speaker and hearer will be most efficient when both parties make accurate inferences about the other. We study inference and communication in a television game called Password, where speakers must convey secret words to hearers by providing one-word clues. Our working hypothesis is that human communication is relatively efficient, and we use game show data to examine three predictions. First, we predict that speakers and hearers are both *considerate*, and that both take the other's perspective into account. Second, we predict that speakers and hearers are *calibrated*, and that both make accurate assumptions about the strategy used by the other. Finally, we predict that speakers and hearers are *collaborative*, and that they tend to share the cognitive burden of communication equally. We find evidence in support of all three predictions, and demonstrate in addition that efficient communication tends to break down when speakers and hearers are placed under time pressure.

## 1 Introduction

Communication and inference are intimately linked. Suppose, for example, that Joan states that some of her pets are dogs. Under normal circumstances, a hearer will infer that not all of Joan's pets are dogs on the grounds that Joan would have expressed herself differently if all of her pets were dogs [1]. Inferences like these have been widely studied by linguists and psychologists [2, 3, 4, 5] and are often encountered in everyday settings. One compelling explanation is presented by Levinson [4], who points out that speaking (i.e. phonetic articulation) is substantially slower than thinking (i.e. inference). As a result, communication will be maximally efficient if a speaker's utterance leaves inferential gaps that will be bridged by the hearer. Inference, however, is not only the responsibility of the hearer. For communication to be maximally efficient, a speaker must take the hearer's perspective into account ("if I say X, will she infer Y?"). The hearer should therefore allow for inferences on the part of the speaker ("did she think that saying X would lead me to infer Y?") Considerations of this sort rapidly lead to a game-theoretic regress, and achieving efficient communication under these circumstances begins to look like a very challenging problem.

Here we study a simple communication game that allows us to explore inferences made by speakers and hearers. Inference becomes especially important in settings where speakers are prevented from directly expressing the concepts they have in mind, and where utterances are constrained to be short. The television show *Password* is organized around a game that satisfies both constraints. In this game, a speaker is supplied with a single, secret word (the password) and must communicate this word to a hearer by choosing a single one-word clue. For example, if the password is "mend", then the speaker might choose "sew" as the clue, and the hearer might guess "stitch" in response. Figure 1 shows several examples drawn from the show—note that communication is successful in the first

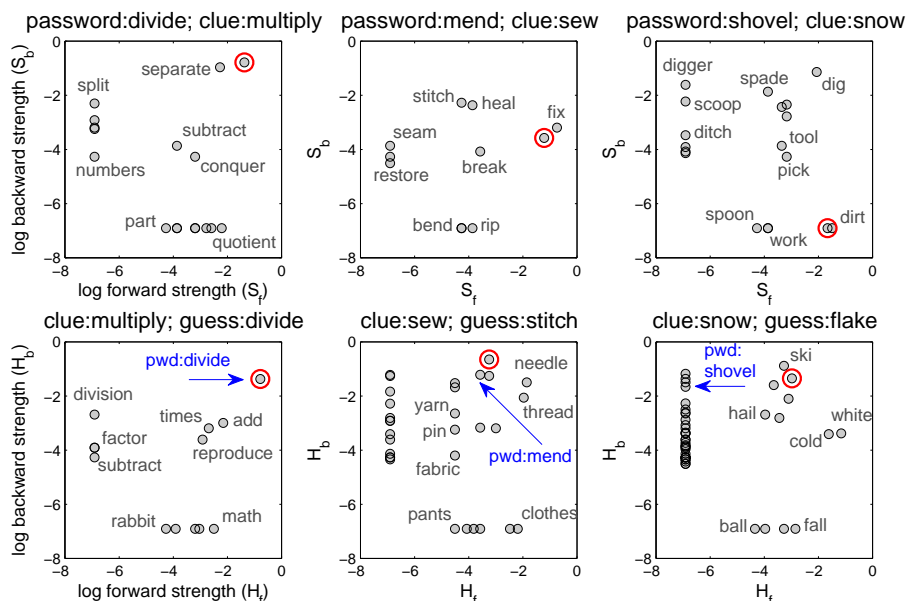

Figure 1: Three rounds from the television game show Password. Given each password, the top row plots the forward ($S_f$: password → clue) and backward ($S_b$: password ← clue) strengths for several potential clues. The clue chosen by the speaker is circled. Given this clue, the bottom row plots the forward ($H_f$: clue → guess) and backward ($H_b$: clue ← guess) strengths for several potential guesses. The guess chosen by the hearer is circled and the password is indicated by an arrow. The first two columns represent two normal rounds, and the final column is a lightning round where speakers and hearers are placed under time pressure. The gray dots in each plot show words that are associated with the password (top row) or clue (bottom row) in the University of Southern Florida word association database. Labels for these words are included where space permits.

example but not in the remaining two. The clues and guesses generated by speakers and hearers are obviously much simpler than most real-world linguistic utterances, but studying a setting this simple allows us to develop and evaluate formal models of communication. Our analyses therefore contribute to a growing body of work that uses formal methods to explore the efficiency of human communication [6, 7, 8, 9, 10, 11, 12, 13, 14, 15, 16].

At first sight the optimal strategies for speaker and hearer may seem obvious: the speaker should generate the clue that is associated most strongly with the password, and the hearer should guess the word that is associated most strongly with the clue. Note, however, that word associations are asymmetric. Given a pair of words such as "shovel" and "snow", the forward association (shovel → snow) may be strong but the backward association (shovel ← snow) may be weak. The third example in Figure 1 shows a case where communication fails because the speaker chooses a clue with a strong forward association but a weak backward association. Although the data include examples like the case just described, we hypothesize that speakers and hearers are both *considerate*: in other words, that both parties attempt to take the other's perspective into account. We test this hypothesis by exploring whether speakers and hearers tend to take backward associations into account when generating their clues and guesses.

Our second hypothesis is that speaker and hearer are *calibrated*: in other words, that both make accurate assumptions about the strategy used by the other. Taking the other person's perspective into account is a good start, but is no guarantee of calibration. Suppose, for example, that the speaker attempts to make the hearer's task as easy as possible, and considers only backward associations when choosing his clue. This strategy will work best if the hearer considers only forward associates of the clue, but suppose that the hearer considers only backward associations, on the theory that the speaker probably generated his clue by choosing a forward associate. In this case, both parties are considerate but not calibrated, and communication is unlikely to prove successful.

Our third hypothesis is that speakers and hearers are *collaborative*: in other words, that they settle on strategies that tend to share the cognitive burden of communication. In operationalizing this hypothesis we assume that forward associates are easier for people to generate than backward associates. A pair of strategies can be calibrated but not cooperative: for example, the speaker and hearer will be calibrated if both agree that the speaker will consider only forward associates, and the hearer will consider only backward associates. This policy, however, is likely to demand more effort from the hearer than the speaker, and we propose that speakers and hearers will satisfy the principle of least collaborative effort [17, 18] by choosing a calibrated pair of strategies where each person weights forward and backward associates equally.

To evaluate our hypotheses we use word association data to analyze the choices made by game show contestants. We first present evidence that speakers and hearers are considerate and take both forward and backward associations into account. We then develop simple models of the speaker and hearer, and use these models to explore the extent to which speakers and hearers weight forward and backward associations. Our results suggest that speakers and hearers are both calibrated and collaborative under normal conditions, but that calibration and collaboration tend to break down under time pressure.

## 2 Game show and word association data

We collected data from the Password game show hosted by Allen Ludden on CBS. Previous researchers have used game show data to explore several aspects of human decision-making [19], but to our knowledge the game of Password has not been previously studied. In each game round, a single English word (the password) is shown to speakers on two competing teams. With each team taking turns, the speaker gives a one-word clue to the hearer and the hearer makes a one-word guess in return. The team that performs best proceeds to the *lightning* rounds where the same game is played under time pressure. Our data set includes passwords, speaker-generated clues and hearer-generated guesses for 100 normal and 100 lightning rounds sampled from the show episodes during 1962–1967. Each round includes a single password and potentially multiple clues and guesses from both teams. For all our our analyses, we use only the first clue–guess pair in each round.

The responses of speakers and hearers are likely to depend heavily on word associations, and we can therefore use word association data to model both speakers and hearers. We used the word association database from the University of South Florida (USF) for all of our analyses [20]. These data were collected using a free association task, where participants were given a cue word and asked to generate a single associate of the cue. More than 6000 participants contributed to the database, and each generated associates for 100–120 English words. To allow for weak associates that were not generated by these participants, we added a count of 1 to the observed frequency for each cue-target pair in the database. The forward strength $(w_i \rightarrow w_j)$ is defined as the proportion of $w_i$ trials where $w_j$ was generated as an associate. The backward strength $(w_i \leftarrow w_j)$ is proportional to the forward strength $(w_j \rightarrow w_i)$ but is normalized with respect to all forward strengths to $w_i$:

$$(w_i \leftarrow w_j) = \frac{(w_j \rightarrow w_i)}{\sum_k (w_k \rightarrow w_i)}. \tag{1}$$

Note that this normalization ensures that both forward and backward strengths can be treated as probabilities. The correlation between forward strengths and backward strengths is positive but low $(r = 0.32)$, suggesting that our game show analyses may be able to differentiate the influence of forward and backward associations.

The USF database includes associates for a set of 5016 words, and we used this set as the lexicon for all of our analyses. Some of the rounds in our game show data include passwords, clues or guesses that do not appear in this lexicon, and we removed these rounds, leaving 68 password-clue and 68 clue-guess pairs in the normal rounds and 86 password-clue pairs and 80 clue-guess pairs in the lightning rounds. The USF database also includes the frequency of each word in a standard corpus of written English [21], and we use these frequencies in our first analysis.

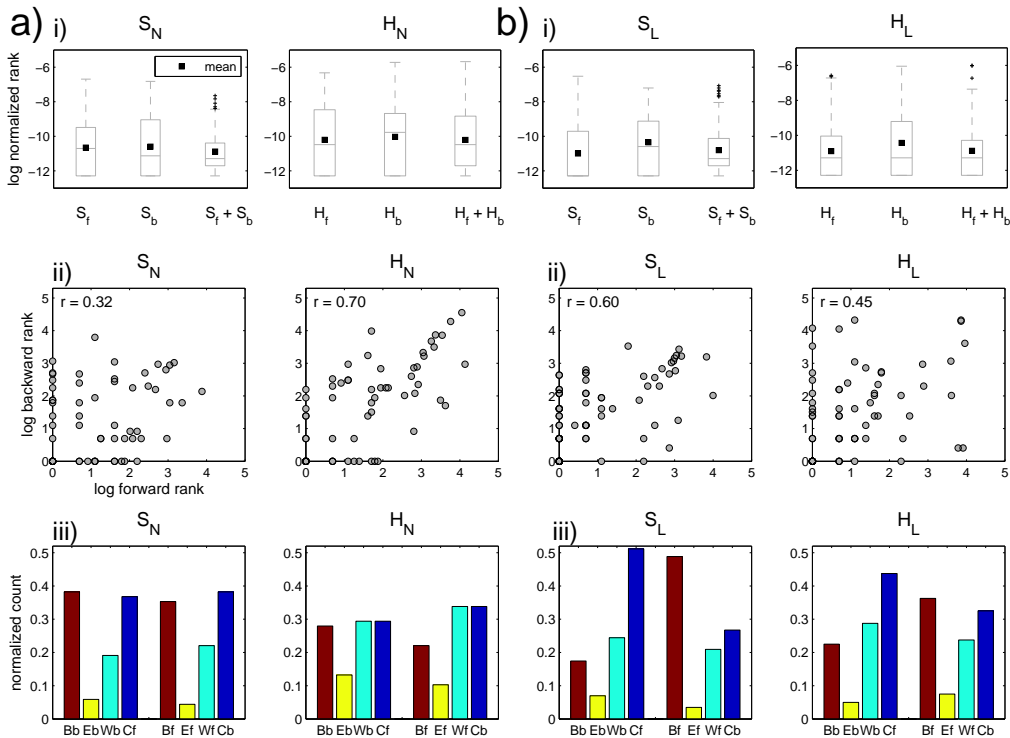

Figure 2: (a) Analyses of the speaker and hearer data ($S_N$ and $H_N$) from the normal rounds. (i) Ranks of the human responses normalized with respect to all other words in the lexicon. Ranks are shown along three dimensions: forward strength ($f$), backward strength ($b$) and combined forward and backward strengths. The dark square shows the mean rank, and the horizontal lines within the box show the median and interquartile range. The plus symbols are outliers. (ii) Ranks of the human responses along the forward and backward dimensions. (iii) "Matched rank" analysis exploring whether human responses tend to be better along one of the dimensions than alternatives that are matched along the other dimension. The four bars on the left in each subplot show normalized counts based on comparisons with matches along the $f$ dimension, and the four bars on the right are based on matches along the $b$ dimension. For example, group $Bb$ includes human responses that are better along the $b$ dimension compared to matches along the $f$ dimension, and groups $Eb$ and $Wb$ include cases where human responses are equal to or worse than the $f$-matches. Group $Cf$ includes cases where the human response is top ranked along the $f$ dimension. Groups $Bf$, $Ef$, $Wf$ and $Cb$ are defined similarly. (b) Analyses of the lightning rounds.

## 3   Speakers and hearers are considerate

A speaker should find it easy to generate clues that are strong forward associates of a password, and a hearer should likewise find it easy to generate guesses that are strong forward associates of a clue. A *considerate* speaker, however, may attempt to generate strong backward associates, which will make it easier for the hearer to successfully guess the password. Similarly, a hearer who considers the task faced by the speaker should also take backward associates into account. This section describes some initial analyses that explore whether clues and guesses are shaped by backward associations.

Figure 2a.i compares forward and backward strengths as predictors of the responses chosen by speakers and hearers. A dimension is a successful predictor if the words chosen by contestants tend to have low ranks along this dimension with respect to the 5016 words in the lexicon (rank 1 is the top rank). We handle ties using fractional ranking, which means that it is sensible to compare mean ranks along each dimension. In Figure 2a.i, $S_f$ and $S_b$ represent forward (password $\rightarrow$ clue) and backward (password $\leftarrow$ clue) strengths for the speaker, and $H_f$ and $H_b$ represent forward (clue $\rightarrow$ guess) and

backward (clue ← guess) strengths for the hearer. In addition to forward and backward strengths, we also considered word frequency as a predictor. Across both normal ($S_N$ and $H_N$) and lightning ($S_L$ and $H_L$) rounds, the ranks along the forward and backward dimensions are substantially better than ranks along the frequency dimension ($p < 0.01$ in pairwise t-tests), and we therefore focus on forward and backward strengths for the rest of our analyses.

For data set $S_N$ the mean ranks suggest that forward and backward strengths appear to predict choices about equally well. The third dimension $S_f + S_b$ is created by combining dimensions $S_f$ and $S_b$. Word $w_1$ dominates $w_2$ if it is superior along one dimension and no worse along the other, and the rank for each word along the combined dimension is based on the number of words that dominate it. For data set $S_N$, the mean rank based on the $S_f + S_b$ dimension is lower than that for $S_f$ alone, suggesting that backward strengths make a predictive contribution that goes beyond the information present in the forward associations. Note, however, that the difference between mean ranks for $S_f$ and $S_f + S_b$ is not statistically significant.

For data set $H_N$, Figure 2a.i provides little evidence that backward strengths make a contribution that goes beyond the forward strengths. Figure 2a.ii plots the rank of each guess along the dimensions of forward and backward strength. The correlation between the dimensions is relatively high, suggesting that both dimensions tend to capture the information present in the other. As a result, the hearer data set $H_N$ may offer little opportunity to explore whether backward and forward associations both contribute to people's responses.

Figure 2a.iii shows the results of an analysis that explores more directly whether each dimension makes a contribution that goes beyond the other. We compared each "actual word" (i.e. each clue or guess chosen by a contestant) to "matched words" that are matched in rank along one of the dimensions. For example, if the backward dimension matters, then the actual words should tend to be better along the $b$ dimension than words that are matched along the $f$ dimension. The first group of bars in Figure 2a.iii shows the proportion of actual words that are better ($Bb$), equivalent ($Eb$) or worse ($Wb$) along the backward dimension than matches along the forward dimension. The $Bb$ bar is higher than the others, suggesting that the backward dimension does indeed make a contribution that goes beyond the forward dimension. Note that a match is defined as a word that is ranked the same as the actual word, or in cases where there are no ties, a word that is ranked one step better. The fourth bar ($Cf$, for champion along the forward dimension) includes all cases where a word is ranked best along the forward dimension, which means that no match can be found. Our policy for identifying matches is conservative—all other things being equal, actual words should be equivalent ($Eb$) or worse ($Wb$) than the matched words, which means that the large $Bb$ bar provides strong evidence that the backward dimension is important. A binomial test confirms that the $Bb$ bar is significantly greater than the $Wb$ bar ($p < 0.05$). The $Bf$ bar for the speaker data is also high, suggesting that the forward dimension makes a contribution that goes beyond the backward dimension. In other words, Figure 2a.iii suggests that both dimensions influence the responses of the speaker.

The results for the hearer data $H_N$ provide additional support for the idea that neither dimension predicts hearer guesses better than the other. Note, for example, that the second group of four bars in Figure 2a.iii suggests that the forward dimension is not predictive once the backward dimension is taken into account ($Bf$ is smaller than $Wf$). This result is consistent with our previous finding that forward and backward strengths are highly correlated in the case of the hearer, and that neither dimension makes a contribution after controlling for the other.

Our analyses so far suggest that forward and backward strengths both make independent contributions to the choices made by speakers, but that the hearer data do not allow us to discriminate between these dimensions. Figure 2b shows similar analyses for the lightning rounds. The most notable change is that backward strengths appear to play a much smaller role when speakers are placed under time pressure. For example, Figure 2b.i suggests that backward strengths are now worse than forward strengths at predicting the clues chosen by speakers. Relative to the results for the normal rounds $S_N$, the $Bb$ counts for $S_L$ in Figure 2b.iii show a substantial drop (53% decrease) and the $Bf$ counts show an increase of similar scale. $\chi^2$ goodness-of-fit tests show that the distributions of counts for both $\{Bb, Eb, Wb, Cf\}$ and $\{Bf, Ef, Wf, Cb\}$ in the lightning rounds significantly deviate from those in the normal rounds ($p < 0.01$). This result provides further evidence that speakers tend to rely more heavily on forward associations than backward associations when placed under time pressure.

| | Speaker distribution $p_S(c\|w)$ | | Hearer distribution $p_H(w\|c)$ |
|---|---|---|---|
| $S_0$ | $(w \to c)$ | $H_0$ | $(c \to w)$ |
| $S_1$ | $(w \leftarrow c)$ | $H_1$ | $(c \leftarrow w)$ |
| $S_2$ | $\alpha_S^{(2)}(w \to c) + \beta_S^{(2)}(w \leftarrow c)$ | $H_2$ | $\alpha_H^{(2)}(c \to w) + \beta_H^{(2)}(c \leftarrow w)$ |
| | $\vdots$ | | $\vdots$ |
| $S_n$ | $\alpha_S^{(n)}(w \to c) + \beta_S^{(n)}(w \leftarrow c)$ | $H_n$ | $\alpha_H^{(n)}(c \to w) + \beta_H^{(n)}(c \leftarrow w)$ |

Table 1: Strategies for speaker and hearer. In each case we assume that the speaker and hearer sample words from distributions $p_S(c|w)$ and $p_H(w|c)$ based on the expressions shown. At level 0, both speaker and hearer rely entirely on forward associates, and at level 1, both parties rely entirely on backward associates. For each party, the strategy at level $k$ is the best choice assuming that the other person uses a strategy at a level lower than $k$.

Our previous analyses found little evidence that forward and backward strengths make separate contributions in the case of the hearer, but the lightning data $H_L$ suggest that these dimensions may indeed make separate contributions. Figure 2b.iii suggests that time pressure affects these dimensions differently: note that $Bb$ counts decrease by 19% and $Bf$ counts increase by 64%. $\chi^2$ tests confirm that the distributions of $\{Bb, Eb, Wb, Cf\}$ and $\{Bf, Ef, Wf, Cb\}$ in the lightning rounds significantly deviate from those in the normal rounds ($p < 0.01$), suggesting that the hearer (like the speaker) tends to rely on forward strengths rather than backward strengths in the lightning rounds.

Taken together, the full set of results in Figure 2 suggests that the responses of speakers and hearers are both shaped by backward associates—in other words, that both parties are considerate of the other person's situation. The evidence in the case of the speaker is relatively strong and all of the analyses we considered suggest that backward associations play a role. The evidence is weaker in the case of the hearer, and only the comparison between normal and lightning rounds suggests that backward associations play some role.

## 4 Efficient communication: calibration and collaboration

Our analyses so far provide some initial evidence that speakers and hearers are both influenced by forward and backward associations. Given this result, we now consider a model that explores how forward and backward associations are combined in generating a response.

### 4.1 Speaker and hearer models

Since both kinds of associations appear to play a role, we explore a simple speaker model which assumes that the clue $c$ chosen for the password $w$ is sampled from a mixture distribution

$$p_S(c|w) = \alpha_S(w \to c) + \beta_S(w \leftarrow c) \tag{2}$$

where $(w \to c)$ indicates the forward strength from $w$ to $c$, $(w \leftarrow c)$ indicates the backward strength from $c$ to $w$, and $\alpha_S$ and $\beta_S$ are mixture weights that sum to 1. The corresponding hearer model assumes that guess $w$ given clue $c$ is sampled from the mixture distribution

$$p_H(w|c) = \alpha_H(c \to w) + \beta_H(c \leftarrow w). \tag{3}$$

Several possible mixture distributions for speaker and hearer are shown in Table 1. For example, the level 0 distributions assume that speaker and hearer both rely entirely on forward associates, and the level 1 distributions assume that both rely entirely on backward associates. By fitting mixture weights to the game show data we can explore the extent to which speaker and hearer rely on forward and backward associations.

The mixture models in Equations 2 and 3 can be derived by assuming that the hearer relies on Bayesian inference. Using Bayes' rule, the hearer distribution $p_H(w|c)$ can be expressed as

$$p_H(w|c) \propto p_S(c|w)p(w). \tag{4}$$

To simplify our analysis we make three assumptions. First, we assume that the prior $p(w)$ in Equation 4 is uniform. Second, we assume that contestants are near-optimal in many respects but that they sample rather than maximize. In other words, we assume that the hearer samples a guess $w$ from the distribution $p_H(w|c)$ in Equation 4, and that the speaker samples a clue from a distribution $p_S(c|w) \propto p_H(w|c)$. Finally, we assume that the normalizing constant in Equation 1 is 1 for all words $w_i$. This assumption seems reasonable since for our smoothed data set the mean value of the normalizing constant is 1 and the standard deviation is 0.04. Our final assumption simplifies matters considerably since it implies that $(w_i \to w_j) = (w_j \leftarrow w_i)$ for all pairs $w_i$ and $w_j$.

Given these assumptions it is straightforward to show that the level 0 strategies in Table 1 are the best responses to the level 1 strategies, and vice versa. For example, if the speaker uses strategy $S_0$ and samples a clue $c$ from the distribution $p_S(c|w) = w \to c$, then Equation 4 suggests that the hearer should sample a guess $w$ from the distribution $p_H(c|w) \propto (w \to c) = (c \leftarrow w)$. Similarly, if the speaker uses the strategy $S_1$ and samples a clue $c$ from the distribution $p_S(c|w) = (w \leftarrow c)$, then Equation 4 suggests that the hearer should sample a guess $w$ from the distribution $p_H(c|w) \propto (w \leftarrow c) = (c \to w)$.

Suppose now that the hearer is uncertain about the strategy used by the speaker. A level 2 hearer assumes that the speaker could use strategy $S_0$ or strategy $S_1$ and assigns prior probabilities of $\beta_H^{(2)}$ and $\alpha_H^{(2)}$ to these speaker strategies. Since $H_1$ is the appropriate response to $S_0$ and $H_0$ is the appropriate response to $S_1$, the level 2 hearer should sample from the distribution

$$
\begin{aligned}
p_H(w|c) &= p(S_1)p_H(w|c, S_1) + p(S_0)p_H(w|c, S_0) \\
&= \alpha_H^{(2)}(c \to w) + \beta_H^{(2)}(c \leftarrow w).
\end{aligned}
\tag{5}
$$

More generally, suppose that a level $n$ hearer assumes that the speaker uses a strategy from the set $\{S_0, S_1, \ldots, S_{n-1}\}$. Since the appropriate response to any one of these strategies is a mixture similar to Equation 5, it follows that strategy $H_n$ is also a mixture of the distributions $(w \to c)$ and $(w \leftarrow c)$. A similar result holds for the speaker, and strategy $S_n$ in Table 1 also takes the form of a mixture distribution. Our Bayesian analysis therefore suggests that efficient speakers and hearers can be characterized by the mixture models in Equations 2 and 3.

Some pairs of mixture models are *calibrated* in the sense that the hearer model is the best choice given the speaker model and vice versa. Equation 4 implies that calibration is achieved when the forward weight for the speaker matches the backward weight for the hearer ($\alpha_S = \beta_H$) and the backward weight for the speaker matches the forward weight for the hearer ($\beta_S = \alpha_H$). If game show contestants achieve efficient communication, then mixture weights fit to their responses should come close to satisfying this calibration condition.

There are many sets of weights that satisfy the calibration condition. For example, calibration is achieved if the speaker uses strategy $S_0$ and the hearer uses strategy $H_1$. If generating backward associates is more difficult than thinking about forward associates, this solution seems unbalanced since the hearer alone is required to think about backward associates. Consistent with the principle of least collaborative effort, we make a second prediction that speaker and hearer will collaborate and share the communicative burden equally. More precisely, we predict that both parties will assign the same weight to backward associates and that $\beta_S$ will equal $\beta_H$. Combining our two predictions, we expect that the weights which best characterize human responses will have $\alpha_S = \beta_S = \alpha_H = \beta_H = 0.5$.

## 4.2 Fitting forward and backward mixture weights to the data

To evaluate our predictions we assumed that the speaker and hearer are characterized by Equations 2 and 3 and identified the mixture weights that best fit the game show data. Assuming that the $M$ game rounds are independent, the log likelihood for the speaker data is

$$
L = \log \prod_{m=1}^{M} P(c^m|w^m) = \sum_{m=1}^{M} [\alpha_S \log(w^m \to c^m) + \beta_S \log(w^m \leftarrow c^m)]
\tag{6}
$$

and a similar expression is used for the hearer data. We fit the weights $\alpha_S$ and $\beta_S$ by maximizing the log likelihood in Equation 6. Since this likelihood term is convex and there is a single free parameter ($\alpha_S + \beta_S = 1$), the global optimum can be found by a simple line search over the range $0 < \alpha_S < 1$.

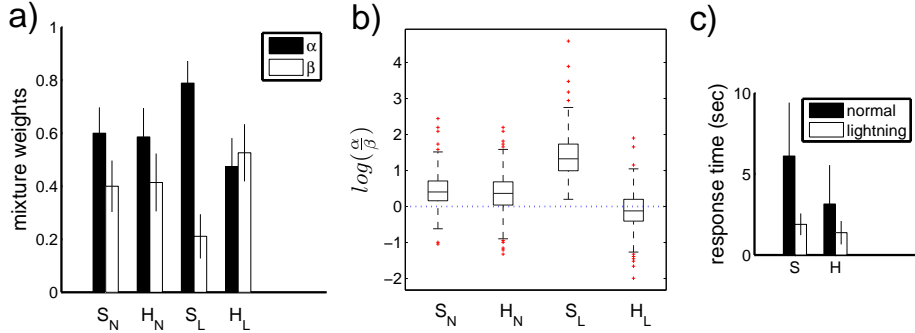

Figure 3: (a) Fitted mixture weights for the speaker ($S$) and hearer ($H$) models based on bootstrapped normal (N) and lightning (L) rounds. $\alpha$ and $\beta$ are weights on the forward and backward strengths. (b) Log-ratios of $\alpha$ and $\beta$ weights estimated from bootstrapped normal and lightning rounds. (c) Average response times for speakers choosing clues and hearers choosing guesses in normal and lightning rounds. Averages are computed over 30 rounds randomly sampled from the game show.

We ran separate analyses for normal and lightning rounds, and ran similar analyses for the hearer data. $1000$ estimates of each mixture weight were computed by bootstrapping game show rounds while keeping tallies of normal and lightning rounds constant.

Consistent with our predictions, the results in Figure 3a suggest that all four mixure weights for the normal rounds are relatively close to $0.5$. Both speaker and hearer appear to weight forward associates slightly more heavily than backward associates, but $0.5$ is within one standard deviation of the bootstrapped estimates in all four cases. The lightning rounds produce a different pattern of results and suggest that the speaker now relies much more heavily on forward than backward associates. Figure 3b shows log ratios of the mixture weights, and indicates that these ratios lie close to $0$ (i.e. $\alpha = \beta$) in all cases except for the speaker in the lightning rounds. Further confidence tests show that the percentage of bootstrapped ratios exceeding $0$ is $100\%$ for the speaker in the lightning rounds, but $85\%$ or lower in the three remaining cases. Consistent with our previous analyses, this result suggests that coordinating with the hearer requires some effort on the part of the speaker, and that this coordination is likely to break down under time pressure. The fitted mixture weights, however, do not confirm the prediction that time pressure makes it difficult for the hearer to consider backward associations. Figure 3c helps to explain why mixture weights for the speaker but not the hearer may differ across normal and lightning rounds. The difference in response times between normal and lightning rounds is substantially greater for the speaker than the hearer, suggesting that any differences between normal and lightning rounds are more likely to emerge for the speaker than the hearer.

## 5 Conclusion

We studied how speakers and hearers communicate in a very simple context. Our results suggest that both parties take the other person's perspective into account, that both parties make accurate assumptions about the strategy used by the other, and that the burden of communication is equally divided between the two. All of these conclusions support the idea that human communication is relatively efficient. Our results, however, suggest that efficient communication is not trivial to achieve, and tends to break down when speakers are placed under time pressure.

Although we worked with simple models of the speaker and hearer, note that neither model is intended to capture psychological processing. Future studies can explore how our models might be implemented by psychologically plausible mechanisms. For example, one possibility is that speakers sample a small set of words with high forward strengths, then choose the word in this sample with greatest backward strength. Different processing models might be considered, but we believe that any successful model of speaker or hearer will need to include some role for inferences about the other person.

**Acknowledgments** This work was supported in part by the Richard King Mellon Foundation (YX) and by NSF grant CDI-0835797 (CK).

# References

[1] L. Horn. Toward a new taxonomy for pragmatic inference: Q-based and R-based implicature. In *Meaning, Form, and Use in Context: Linguistic Applications*. Georgetown University Press, 1984.

[2] P. Grice. *Studies in the Way of Words*. Harvard University Press, Cambridge, 1989.

[3] D. Sperber. *Relevance: Communication and Cognition*. Blackwell, Oxford, 1986.

[4] S. Levinson. *Presumptive Meanings: The Theory of Generalized Implicature*. MIT Press, Cambridge, 2000.

[5] D. Jurafsky. Pragmatics and computational linguistics. In L. R. Horn and G. Ward, editors, *Handbook of Pragmatics*, pages 578–604. Blackwell, Oxford, 2005.

[6] G. K. Zipf, editor. *Human behaviour and the principle of least effort: An introduction to human ecology*. Addison-Wesley Press, Cambridge, 1949.

[7] R. Levy and T. F. Jaeger. Speakers optimize information density through syntactic reduction. In *Advances in Neural Information Processing Systems*, 2007.

[8] T. F. Jaeger. Redundancy and reduction: Speakers manage syntactic information density. *Cognitive Psychology*, 61(1):23–62, 2010.

[9] M. Aylett and A. Turk. The smooth signal redundancy hypothesis: A functional explanation for relationships between redundancy, prosodic prominence, and duration in spontaneous speech. *Language and Speech*, 47(1):31–56, 2004.

[10] S. T. Piantadosi, H. J. Tily, and E. Gibson. The communicative lexicon hypothesis. In *The 31st annual meeting of the Cognitive Science Society*, 2009.

[11] R. Baddeley and D. Attewell. The relationship between language and the environment: information theory shows why we have only three lightness terms. *Psychological Science*, 20(9):1100–1107, 2009.

[12] J. Hawkins. *Efficiency and complexity in grammars*. Oxford University Press, Oxford, 2004.

[13] N. Chomsky. Language and mind: current thoughts on ancient problems. In L. Jenkins, editor, *Variations and universals in biolinguistics*, pages 379–405. Elsevier, Amsterdam, 2004.

[14] R. van Rooy. Conversational implicatures and communication theory. In J. van Kuppevelt and R. Smith, editors, *Current and New Directions in Discourse and Dialogue*. Kluwer, 2003.

[15] C. R. M. McKenzie and J. D. Nelson. What a speaker's choice of frame reveals: reference points, frame selection, and framing effects. *Psychonomic Bullentin and Review*, 10, 2003.

[16] S. Sher and C. R. M. McKenzie. Information leakage from logically equivalent frames. *Cognition*, 101:467–494, 2006.

[17] H. H. Clark and D. Wilkes-Gibbs. Referring as a collaborative process. *Cognition*, 22:1–39, 1986.

[18] H. H. Clark. *Using language*. Cambridge University Press, Cambridge, 1996.

[19] J. B. Berk, E. Hughson, and K. Vandezande. The price is right, but are the bids? An investigation of rational decision theory. *The American Economic Review*, 86(4):654–970, 1996.

[20] D. L. Nelson, C. L. McEvoy, and T. A. Schreiber. The University of South Florida word association, rhyme, and word fragment norms. http://www.usf.edu/FreeAssociation/, 1998.

[21] H. Kucera and W. N. Francis. *Computational Analysis of Present-day American Engish*. Brown University Press, Providence, 1967.

